# Non-conjugate Variational Message Passing for Multinomial and Binary Regression

**David A. Knowles**
Department of Engineering
University of Cambridge

**Thomas P. Minka**
Microsoft Research
Cambridge, UK

## Abstract

Variational Message Passing (VMP) is an algorithmic implementation of the Variational Bayes (VB) method which applies only in the special case of conjugate exponential family models. We propose an extension to VMP, which we refer to as Non-conjugate Variational Message Passing (NCVMP) which aims to alleviate this restriction while maintaining modularity, allowing choice in how expectations are calculated, and integrating into an existing message-passing framework: Infer.NET. We demonstrate NCVMP on logistic binary and multinomial regression. In the multinomial case we introduce a novel variational bound for the softmax factor which is tighter than other commonly used bounds whilst maintaining computational tractability.

## 1 Introduction

Variational Message Passing [20] is a message passing implementation of the mean-field approximation [1, 2], also known as variational Bayes (VB). Although Expectation Propagation [12] can have more desirable properties as a result of the particular Kullback-Leibler divergence that is minimised, VMP is more stable than EP under certain circumstances, such as multi-modality in the posterior distribution.

Unfortunately, VMP is effectively limited to conjugate-exponential models since otherwise the messages become exponentially more complex at each iteration. In conjugate exponential models this is avoided due to the closure of exponential family distributions under multiplication. There are many non-conjugate problems which arise in Bayesian statistics, for example logistic regression or learning the hyperparameters of a Dirichlet.

Previous work extending Variational Bayes to non-conjugate models has focused on two aspects. The first is how to fit the variational parameters when the VB free form updates are not viable. Various authors have used standard numerical optimization techniques [15, 17, 3], or adapted such methods to be more suitable for this problem [7, 8]. A disadvantage of this approach is that the convenient and efficient message-passing formulation is lost.

The second line of work applying VB to non-conjugate models involves deriving lower bounds to approximate the expectations [9, 18, 5, 10, 11] required to calculate the KL divergence. We contribute to this line of work by proposing and evaluating a new bound for the useful softmax factor, which is tighter than other commonly used bounds whilst maintaining computational tractability. We also demonstrate, in agreement with [19] and [14], that for univariate expectations such as required for logistic regression, carefully designed quadrature methods can be effective.

Existing methods typically represent a compromise on modularity or performance. To maintain modularity one is effectively constrained to use exponential family bounds (e.g. quadratic in the Gaussian case [9, 5]) which we will show often gives sub-optimal performance. Methods which uses more general bounds, e.g. [3], must then resort to numerical optimisation, and sacrifice modularity.

This is a particular disadvantage for an inference framework such as Infer.NET [13] where we want to allow modular construction of inference algorithms from arbitrary deterministic and stochastic factors. We propose a novel message passing algorithm, which we call Non-conjugate Variational Message Passing (NCVMP), which generalises VMP and gives a recipe for calculating messages out of any factor. NCVMP gives much greater freedom in how expectations are taken (using bounds or quadrature) so that performance can be maintained along with modularity.

The outline of the paper is as follows. In Sections 2 and 3 we briefly review VB and VMP. Section 4 is the main contribution of the paper: Non-conjugate VMP. Section 5 describes the binary logistic and multinomial softmax regression models, and implementation options with and without NCVMP. Results on synthetic and standard UCI datasets are given in Section 6 and some conclusions are drawn in Section 7.

## 2 Mean-field approximation

Our aim is to approximate some model $p(\mathbf{x})$, represented as a factor graph $p(\mathbf{x}) = \prod_a f_a(\mathbf{x}_a)$ where factor $f_a$ is a function of all $x \in \mathbf{x}_a$. The mean-field approximation assumes a fully-factorised variational posterior $q(\mathbf{x}) = \prod_i q_i(x_i)$ where $q_i(x_i)$ is an approximation to the marginal distribution of $x_i$ (note however $x_i$ might be vector valued, e.g. with multivariate normal $q_i$). The variational approximation $q(\mathbf{x})$ is chosen to minimise the Kullback-Leibler divergence $KL(q||p)$, given by

$$KL(q||p) = \int q(\mathbf{x}) \log \frac{q(\mathbf{x})}{p(\mathbf{x})} d\mathbf{x} = -H[q(\mathbf{x})] - \int q(\mathbf{x}) \log p(\mathbf{x}) d\mathbf{x}. \tag{1}$$

where $H[q(\mathbf{x})] = -\int q(\mathbf{x}) \log q(\mathbf{x}) d\mathbf{x}$ is the entropy. It can be shown [1] that if the functions $q_i(x_i)$ are *unconstrained* then minimising this functional can be achieved by coordinate descent, setting $q_i(x_i) = \exp\langle \log p(\mathbf{x})\rangle_{\neg q_i(x_i)}$, iteratively for each $i$, where $\langle ...\rangle_{\neg q_i(x_i)}$ implies marginalisation of all variables except $x_i$.

## 3 Variational Message Passing on factor graphs

VMP is an efficient algorithmic implementation of the mean-field approximation which leverages the fact that the mean-field updates only requires local operations on the factor graph. The variational distribution $q(\mathbf{x})$ factorises into approximate factors $\tilde{f}_a(\mathbf{x}_a)$. As a result of the fully factorised approximation, the approximate factors themselves factorise into messages, i.e. $\tilde{f}_a(\mathbf{x}_a) = \prod_{x_i \in \mathbf{x}_a} m_{a \to i}(x_i)$ where the message from factor $a$ to variable $i$ is $m_{a \to i}(x_i) = \exp\langle \log f_a(\mathbf{x}_a)\rangle_{\neg q_i(x_i)}$. The message from variable $i$ to factor $a$ is the current variational posterior of $x_i$, denoted $q_i(x_i)$, i.e. $m_{i \to a}(x_i) = q_i(x_i) = \prod_{a \in \mathcal{N}(i)} m_{a \to i}(x_i)$ where $\mathcal{N}(i)$ are the factors connected to variable $i$.

For conjugate-exponential models the messages to a particular variable $x_i$, will all be in the same exponential family. Thus calculating $q_i(x_i)$ simply involves summing sufficient statistics. If, however, our model is not conjugate-exponential, there will be a variable $x_i$ which receives incoming messages which are in different exponential families, or which are not even exponential family distributions at all. Thus $q_i(x_i)$ will be some more complex distribution. Computing the required expectations becomes more involved, and worse still the complexity of the messages (e.g. the number of possible modes) grows exponentially per iteration.

## 4 Non-conjugate Variational Message Passing

In this section we give some criteria under which the algorithm was conceived. We set up required notation and describe the algorithm, and prove some important properties. Finally we give some intuition about what the algorithm is doing. The approach we take aims to fulfill certain criteria:

1. provides a recipe for any factor
2. reduces to standard VMP in the case of conjugate exponential factors
3. allows modular implementation and combining of deterministic and stochastic factors

NCVMP ensures the gradients of the approximate KL divergence implied by the message match the gradients of the true KL. This means that we will have a fixed point at the correct point in parameter space: the algorithm will be at a fixed point if the gradient of the KL is zero.

We use the following notation: variable $x_i$ has current variational posterior $q_i(x_i; \theta_i)$, where $\theta_i$ is the vector of natural parameters of the exponential family distribution $q_i$. Each factor $f_a$ which is a neighbour of $x_i$ sends a message $m_{a \to i}(x_i; \phi_{a \to i})$ to $x_i$, where $m_{a \to i}$ is in the same exponential family as $q_i$, i.e. $m_{a \to i}(x_i; \phi) = \exp(\phi^T \mathbf{u}(x_i) - \kappa(\phi))$ and $q_i(x_i; \theta) = \exp(\theta^T \mathbf{u}(x_i) - \kappa(\theta))$ where $\mathbf{u}(\cdot)$ are sufficient statistics, and $\kappa(\cdot)$ is the log partition function. We define $C(\theta)$ as the Hessian of $\kappa(\cdot)$ evaluated at $\theta$, i.e. $C_{ij}(\theta) = \frac{\partial^2 \kappa(\theta)}{\partial \theta_i \partial \theta_j}$. It is straightforward to show that $C(\theta) = \mathrm{cov}(\mathbf{u}(x)|\theta)$ so if the exponential family $q_i$ is identifiable, $C$ will be symmetric positive definite, and therefore invertible. The factor $f_a$ contributes a term $S_a(\theta_i) = \int q_i(x_i; \theta_i) \langle \log f_a(\mathbf{x}) \rangle_{\neg q_i(x_i)} dx_i$ to the KL divergence, where we have only made the dependence on $\theta_i$ explicit: this term is also a function of the variational parameters of the other variables neighbouring $f_a$. With this notation in place we are now able to describe the NCVMP algorithm.

---

**Algorithm 1** Non-conjugate Variational Message Passing

1: Initialise all variables to uniform $\theta_i := 0 \forall i$
2: **while** not converged **do**
3:     **for** all variables $i$ **do**
4:        **for** all neighbouring factors $a \in \mathcal{N}(i)$ **do**
5:           $\phi_{a \to i} := C(\theta_i)^{-1} \frac{\partial S_a(\theta_i)}{\partial \theta_i}$
6:        **end for**
7:        $\theta_i := \sum_{a \in \mathcal{N}(i)} \phi_{a \to i}$
8:     **end for**
9: **end while**

---

To motivate Algorithm 1 we give a rough proof that we will have a fixed point at the correct point in parameter space: the algorithm will be at a fixed point if the gradient of the KL divergence is zero.

**Theorem 1.** *Algorithm 1 has a fixed point at $\{\theta_i : \forall i\}$ if and only if $\{\theta_i : \forall i\}$ is a stationary point of the KL divergence $KL(q||p)$.*

*Proof.* Firstly define the function

$$\tilde{S}_a(\theta; \phi) := \int q_i(x_i; \theta) \log m_{a \to i}(x_i; \phi) dx_i, \qquad (2)$$

which is an approximation to the function $S_a(\theta)$. Since $q_i$ and $m_{a \to i}$ belong to the same exponential family we can simplify as follows,

$$\tilde{S}_a(\theta; \phi) = \int q_i(x_i; \theta)(\phi^T \mathbf{u}(x_i) - \kappa(\phi)) dx_i = \phi^T \langle \mathbf{u}(x_i) \rangle_\theta - \kappa(\phi) = \phi^T \frac{\partial \kappa(\theta)}{\partial \theta} - \kappa(\phi), \quad (3)$$

where $\langle \cdot \rangle_\theta$ implies expectation wrt $q_i(x_i; \theta)$ and we have used the standard property of exponential families that $\langle \mathbf{u}(x_i) \rangle_\theta = \frac{\partial \kappa(\theta)}{\partial \theta}$. Taking derivatives wrt $\theta$ we have $\frac{\partial \tilde{S}_a(\theta; \phi)}{\partial \theta} = C(\theta)\phi$. Now, the update in Algorithm 1, Line 5 for $\phi_{a \to i}$ ensures that

$$C(\theta)\phi = \frac{\partial S_a(\theta)}{\partial \theta} \quad \Leftrightarrow \quad \frac{\partial \tilde{S}_a(\theta; \phi)}{\partial \theta} = \frac{\partial S_a(\theta)}{\partial \theta}. \qquad (4)$$

Thus this update ensures that the gradients wrt $\theta_i$ of $S$ and $\tilde{S}$ match. The update in Algorithm 1, Line 7 for $\theta_i$ is minimising an *approximate* local KL divergence for $x_i$:

$$\theta_i := \arg\min_\theta \left( -H[q_i(x_i, \theta)] - \sum_{a \in \mathcal{N}(i)} \tilde{S}(\theta; \phi_{a \to i}) \right) = \sum_{a \in \mathcal{N}(i)} \phi_{a \to i} \qquad (5)$$

where $H[.]$ is the entropy. If and only if we are at a fixed point of the algorithm, we will have

$$\frac{\partial}{\partial \theta_i} \left( -H[q_i(x_i, \theta_i)] - \sum_{a \in \mathcal{N}(i)} \tilde{S}_a(\theta_i; \phi_{a \to i}) \right) = \frac{\partial H[q_i(x_i, \theta_i)]}{\partial \theta_i} - \sum_{a \in \mathcal{N}(i)} \frac{\partial \tilde{S}_a(\theta_i; \phi_{a \to i})}{\partial \theta_i} = 0$$

for all variables $i$. By (4), if and only if we are at a fixed point (so that $\theta_i$ has not changed since updating $\phi$) we have

$$-\frac{\partial H[q_i(x_i, \theta_i)]}{\partial \theta_i} - \sum_{a \in \mathcal{N}(i)} \frac{\partial S_a(\theta_i)}{\partial \theta_i} = \frac{\partial KL(q||p)}{\partial \theta_i} = 0 \tag{6}$$

for all variables $i$. $\square$

Theorem 1 showed that if NCVMP converges to a fixed point then it is at a stationary point of the KL divergence $KL(q||p)$. In practice this point will be a minimum because any maximum would represent an unstable equilibrium. However, unlike VMP we have no guarantee to decrease $KL(q||p)$ at every step, and indeed we do sometimes encounter convergence problems which require damping to fix: see Section 7. Theorem 1 also gives some intuition about what NCVMP is doing. $\tilde{S}_a$ is a conjugate approximation to the true $S_a$ function, chosen to have the correct gradients at the current $\theta_i$. The update at variable $x_i$ for $\theta_i$ combines all these approximations from factors involving $x_i$ to get an approximation to the local KL, and then moves $\theta_i$ to the minimum of this approximation.

Another important property of Non-conjugate VMP is that it reduces to standard VMP for conjugate factors.

**Theorem 2.** *If* $\langle \log f_a(\mathbf{x}) \rangle_{\neg q_i(x_i)}$ *as a function of* $x_i$ *can be written* $\mu^T \mathbf{u}(x_i) - c$ *where c is a constant, then the NCVMP message* $m_{a \rightarrow i}(x_i, \phi_{a \rightarrow i})$ *will be the standard VMP message* $m_{a \rightarrow i}(x_i, \mu)$.

*Proof.* To see this note that $\langle \log f_a(\mathbf{x}) \rangle_{\neg q_i(x_i)} = \mu^T \mathbf{u}(x_i) - c \quad \Rightarrow \quad S_a(\theta) = \mu^T \langle \mathbf{u}(x_i) \rangle_\theta - c$, where $\mu$ is the expected natural statistic under the messages from the variables connected to $f_a$ other than $x_i$. We have $S_a(\theta) = \mu^T \frac{\partial \kappa(\theta)}{\partial \theta} - c \quad \Rightarrow \quad \frac{\partial S_a(\theta)}{\partial \theta} = C(\theta)\mu$ so from Algorithm 1, Line 7 we have $\phi_{a \rightarrow i} := C(\theta)^{-1} \frac{\partial S_a(\theta)}{\partial \theta} = C(\theta)^{-1} C(\theta)\mu = \mu$, the standard VMP message. $\square$

The update for $\theta_i$ in Algorithm 1, Line 7 is the same as for VMP, and Theorem 2 shows that for conjugate factors the messages sent to the variables are the same as for VMP. Thus NCVMP is a generalisation of VMP.

NCVMP can alternatively be derived by assuming the incoming messages to $x_i$ are fixed apart from $m_{a \rightarrow i}(x_i; \phi)$ and calculating a fixed point update for $m_{a \rightarrow i}(x_i; \phi)$. Gradient matching for NCVMP can be seen as analogous to moment matching in EP. Due to space limitations we defer the details to the supplementary material.

### 4.1 Gaussian variational distribution

Here we describe the NCVMP updates for a Gaussian variational distribution $q(x) = N(x; m, v)$ and approximate factor $\tilde{f}(x; m_f, v_f)$. Although these can be derived from the generic formula using natural parameters it is mathematically more convenient to use the mean and variance (NCVMP is parameterisation invariant so it is valid to do this).

$$\frac{1}{v_f} = -2\frac{dS(m, v)}{dv}, \qquad \frac{m_f}{v_f} = \frac{m}{v_f} + \frac{dS(m, v)}{dm}. \tag{7}$$

## 5 Logistic regression models

We illustrate NCVMP on Bayesian binary and multinomial logistic regression. The regression part of the model is standard: $g_{kn} = \sum_{d=1}^{D} W_{kd} X_{dn} + m_k$ where $g$ is the auxiliary variable, $W$ is a matrix of weights with standard normal prior, $X$ is the design matrix and $m$ is a per class mean, which is also given a standard normal prior. For binary regression we just have $k = 1$, and the observation model is $p(y = 1|g_{1n}) = \sigma(g_{1n})$ where $\sigma(x) = 1/(1 + e^{-x})$ is the logistic function. In the multinomial case $p(y = k|g_{:n}) = \sigma_k(g_{:n})$ where $\sigma_k(\mathbf{x}) = \frac{e^{x_k}}{\sum_l e^{x_l}}$ is the "softmax" function. The VMP messages for the regression part of the model are standard so we omit the details due to space limitations.

## 5.1 Binary logistic regression

For logistic regression we require the following factor: $f(s,x) = \sigma(x)^s (1 - \sigma(x))^{1-s}$ where we assume $s$ is observed. The log factor is $sx - \log(1 + e^x)$. There are two problems: we cannot analytically compute expectations wrt to $x$, and we need to optimise the variational parameters. [9] propose the "quadratic" bound on the integrand

$$\sigma(x) \geq \tilde{\sigma}(x,t) = \sigma(t) \exp\left((x-t)/2 - \frac{\lambda(t)}{2}(x^2 - t^2)\right), \tag{8}$$

where $\lambda(t) = \frac{\tanh(t/2)}{t} = \frac{\sigma(t) - 1/2}{t}$. It is straightforward to analytically optimise $t$ to make the bound as tight as possible. The bound is conjugate to a Gaussian, but its performance can be poor. An alternative proposed in [18] is to bound the integral:

$$\langle \log f(x) \rangle_q \geq sm - \frac{1}{2}a^2 v - \log(1 + e^{m+(1-2a)v/2}), \tag{9}$$

where $m, v$ are the mean and variance of $q(x)$ and $a$ is a variational parameter which can be optimised using the fixed point iteration $a := \sigma(m - (1-2a)v/2)$. We refer to this as the "tilted" bound. This bound is not conjugate to a Gaussian, but we can calculate the NCVMP message, which has parameters: $\frac{1}{v_f} = a(1-a), \frac{m_f}{v_f} = \frac{m}{v_f} + s - a$, where we have assumed $a$ has been optimised. A final possibility is to use quadrature to calculate the gradients of $S(m,v)$ directly. The NCVMP message then has parameters $\frac{1}{v_f} = \frac{\langle x\sigma(x)\rangle_q - m\langle\sigma(x)\rangle_q}{v}, \frac{m_f}{v_f} = \frac{m}{v_f} + s - \langle\sigma(x)\rangle_q$. The univariate expectations $\langle\sigma(x)\rangle_q$ and $\langle x\sigma(x)\rangle_q$ can be efficiently computed using Gauss-Hermite or Clenshaw-Curtis quadrature.

## 5.2 Multinomial softmax regression

Consider the softmax factor $f(x,p) = \prod_{k=1}^{K} \delta(p_k - \sigma_k(\mathbf{x}))$, where $x_k$ are real valued and $p$ is a probability vector with current Dirichlet variational posterior $q(p) = \text{Dir}(p;d)$. We can integrate out $p$ to give the log factor $\log f(x) = \sum_{k=1}^{K}(d_k - 1)x_k - (d_. - K)\log\sum_l e^{x_l}$ where we define $d_. := \sum_{k=1}^{K} d_k$. Let the incoming message from $x$ be $q(x) = \prod_{k=1}^{K} N(x_k; m_k, v_k)$. How should we deal with the $\log\sum_l e^{x_l}$ term? The approach used by [3] is a linear Taylor expansion of the log, which is accurate for small variances $v$:

$$\langle \log \sum_i e^{x_i} \rangle \leq \log \sum_i \langle e^{x_i} \rangle = \log \sum_i e^{m_i + v_i/2}, \tag{10}$$

which we refer to as the "log" bound. The messages are still not conjugate, so some numerical method must still be used to learn $m$ and $v$: while [3] used LBFGS we will use NCVMP. Another bound was proposed by [5]:

$$\log \sum_{k=1}^{K} e^{x_k} \leq a + \sum_{k=1}^{K} \log(1 + e^{x_k - a}), \tag{11}$$

where $a$ is a new variational parameter. Combining with (8) we get the "quadratic bound" on the integrand, with $K + 1$ variational parameters. This has conjugate updates, so modularity can be achieved without NCVMP, but as we will see, results are often poor. [5] derives coordinate ascent fixed point updates to optimise $a$, but reducing to a univariate optimisation in $a$ and using Newton's method is much faster (see supplementary material).

Inspired by the univariate "tilted" bound in Equation 9 we propose the multivariate tilted bound:

$$\langle \log \sum_i e^{x_i} \rangle \leq \frac{1}{2}\sum_j a_j^2 v_j + \log \sum_i e^{m_i + (1-2a_i)v_i/2} \tag{12}$$

Setting $a_k = 0$ for all $k$ we recover Equation 10 (hence this is the "tilted" version). Maximisation with respect to $\mathbf{a}$ can be achieved by the fixed point update (see supplementary material): $\mathbf{a} := \sigma\left[\mathbf{m} + \frac{1}{2}(\mathbf{1} - 2\mathbf{a}) \cdot \mathbf{v}\right]$. This is a $\mathcal{O}(K)$ operation since the denominator of the softmax function is shared. For the softmax factor quadrature is not viable because of the high dimensionality of the integrals. From Equation 7 the NCVMP messages using the tilted bound have natural parameters

$\frac{1}{v_{kf}} = (d_{.} - K)a_k(1 - a_k), \frac{m_{kf}}{v_{kf}} = \frac{m_k}{v_{kf}} + d_k - 1 - (d_{.} - K)a_k$ where we have assumed $\mathbf{a}$ has been optimised. As an alternative we suggest choosing whether to send the message resulting from the quadratic bound or tilted bound depending on which is currently the tightest, referred to as the "adaptive" method. Finally we consider a simple Taylor series expansion of the integrand around the mean of $x$, denoted "Taylor", and the multivariate quadratic bound of [4], denoted "Bohning" (see the Supplementary material for details).

## 6   Results

Here we aim to present the typical compromise between performance and modularity that NCVMP addresses. We will see that for both binary logistic and multinomial softmax models achieving conjugate updates by being constrained to quadratic bounds is sub-optimal, in terms of estimates of variational parameters, marginal likelihood estimation, and predictive performance. NCVMP gives the freedom to choose a wider class of bounds, or even use efficient quadrature methods in the univariate case, while maintaining simplicity and modularity.

### 6.1   The logistic factor

We first test the logistic factor methods of Section 5.1 at the task of estimating the toy model $\sigma(x)\pi(x)$ with varying Gaussian prior $\pi(x)$ (see Figure 1(a)). We calculate the true mean and variance using quadrature. The quadratic bound has the largest errors for the posterior mean, and the posterior variance is severely underestimated. In contrast, NCVMP using quadrature, while being slightly more computationally costly, approximates the posterior much more accurately: the error here is due only to the VB approximation. Using the tilted bound with NCVMP gives more robust estimates of the variance than the quadratic bound as the prior mean changes. However, both the quadratic and tilted bounds underestimate the variance as the prior variance increases.

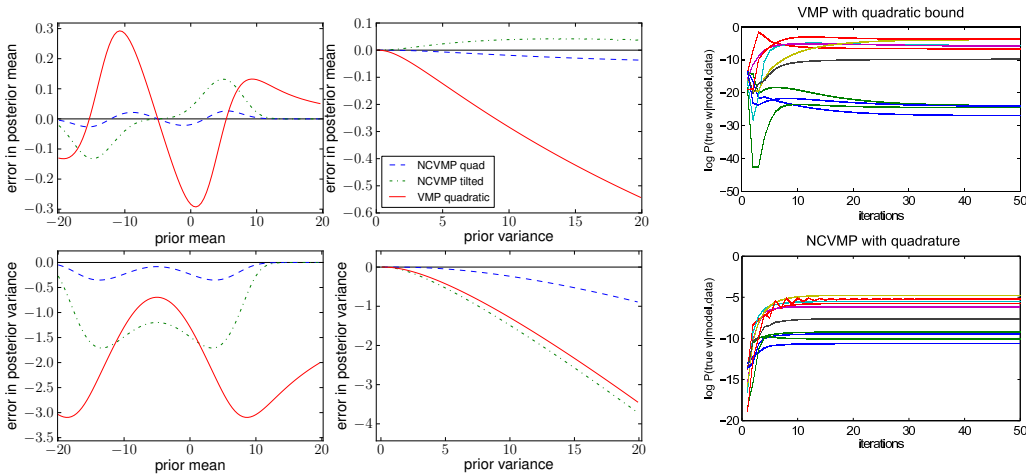

(a) Posterior mean and variance estimates of $\sigma(x)\pi(x)$ with varying prior $\pi(x)$. Left: varying the prior mean with fixed prior variance $v = 10$. Right: varying the prior variance with fixed prior mean $m = 0$.

(b) Log likelihood of the true regression coefficients under the approximate posterior for 10 synthetic logistic regression datasets.

Figure 1: Logistic regression experiments.

### 6.2   Binary logistic regression

We generated ten synthetic logistic regression datasets with $N = 30$ data points and $P = 8$ covariates. We evaluated the results in terms of the log likelihood of the true regression coefficients under the approximate posterior, a measure which penalises poorly estimated posterior variances. Figure 1(b) compares the performance of non-conjugate VMP using quadrature and VMP using the quadratic bound. For four of the ten datasets the quadratic bound finds very poor solutions. Non-conjugate VMP finds a better solution in seven out of the ten datasets, and there is marginal

difference in the other three. Non-conjugate VMP (with no damping) also converges faster in general, although some oscillation is seen for one of the datasets.

## 6.3 Softmax bounds

To have some idea how the various bounds for the softmax integral $\mathbb{E}_q[\log \sum_{k=1}^{K} e^{x_k}]$ compare empirically we calculated relative absolute error on 100 random distributions $q(x) = \prod_k N(x_k; m_k, v)$. We sample $m_k \sim N(0, u)$. When not being varied, $K = 10, u = 1, v = 1$. Ground truth was calculated using $10^5$ Monte Carlo samples. We vary the number of classes, $K$, the distribution variance $v$ and spread of the means $u$. Results are shown in Figure 2. As expected the tilted bound (12) dominates the log bound (10), since it is a generalisation. As $K$ is increased the relative error made using the quadratic bound increases, whereas both the log and the tilted bound get tighter. In agreement with [5] we find the strength of the quadratic bound (11) is in the high variance case, and Bohning's bound [4] is very loose under all conditions. Both the log and tilted bound are extremely accurate for variances $v < 1$. In fact, the log and tilted bounds are asymptotically optimal as $v \to 0$. "Taylor" gives accurate results but is not a bound, so convergence is not guaranteed and the global bound on the marginal likelihood is lost. The spread of the means $u$ does not have much of an effect on the tightness of these bounds. These results show that even when quadrature is not an option, much tighter bounds can be found if the constraint of requiring quadratic bounds imposed by VMP is relaxed. For the remainder of the paper we consider only the quadratic, log and tilted bounds.

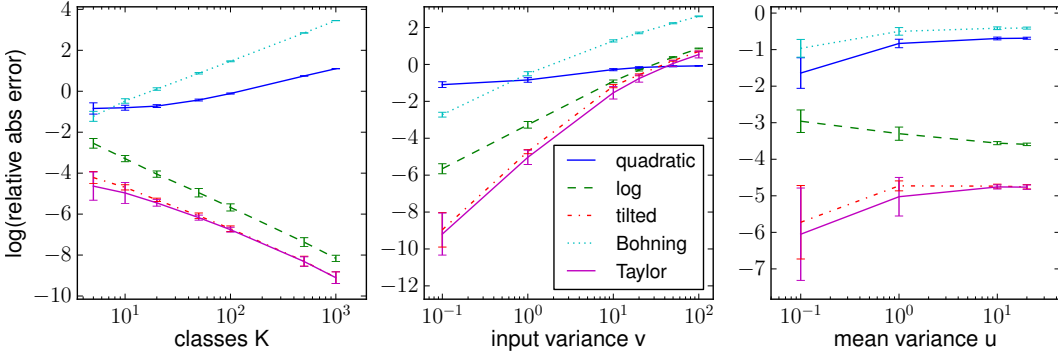

Figure 2: Log10 of the relative absolute error approximating $\mathbb{E} \log \sum \exp$, averaged over 100 runs.

## 6.4 Multinomial softmax regression

**Synthetic data.** For synthetic data sampled from the generative model we know the ground truth coefficients and can control characteristics of the data. We first investigate the performance with sample size $N$, with fixed number of features $P = 6$, classes $K = 4$, and no noise (apart from the inherent noise of the softmax function). As expected our ability to recover the ground truth regression coefficients improves with increasing $N$ (see Figure 3(a), left). However, we see that the methods using the tilted bound perform best, closely followed by the log bound. Although the quadratic bound has comparable performance for small $N < 200$ it performs poorly with larger $N$ due to its weakness at small variances. The choice of bound impacts the speed of convergence (see Figure 3(a), right). The log bound performed almost as well as the tilted bound at recovering coefficients it takes many more iterations to converge. The extra flexibility of the tilted bound allows faster convergence, analogous to parameter expansion [16]. For small $N$ the tilted bound, log bound and adaptive method converge rapidly, but as $N$ increases the quadratic bound starts to converge much more slowly, as do the tilted and adaptive methods to a lesser extent. "Adaptive" converges fastest because the quadratic bound gives good initial updates at high variance, and the tilted bound takes over once the variance decreases. We vary the level of noise in the synthetic data, fixing $N = 200$, in Figure 3(b). For all but very large noise values the tilted bound performs best.

**UCI datasets.** We test the multinomial regression model on three standard UCI datasets: Iris ($N = 150, D = 4, K = 3$), Glass ($N = 214, D = 8, K = 6$) and Thyroid ($N = 7200, D = 21, K = 3$),

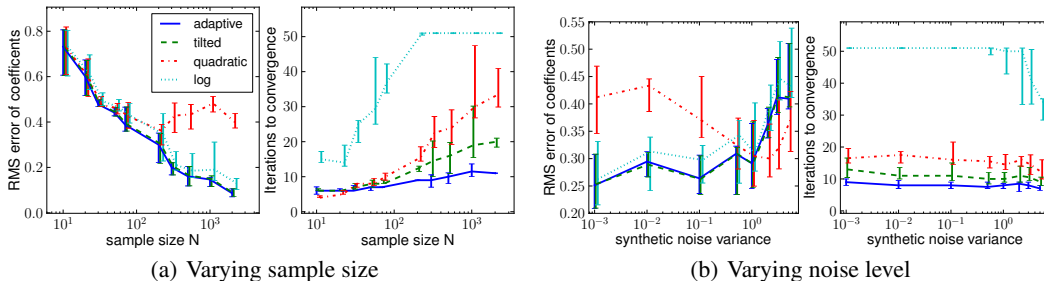

(a) Varying sample size          (b) Varying noise level

Figure 3: Left: root mean squared error of inferred regression coefficients. Right: iterations to convergence. Results are shown as quartiles on 16 random synthetic datasets. All the bounds except "quadratic" were fit using NCVMP.

| Iris | Quadratic | Adaptive | Tilted | Probit |
|---|---|---|---|---|
| Marginal likelihood | $-65 \pm 3.5$ | $-31.2 \pm 2$ | $-31.2 \pm 2$ | $-37.3 \pm 0.79$ |
| Predictive likelihood | $-0.216 \pm 0.07$ | $-0.201 \pm 0.039$ | $-0.201 \pm 0.039$ | $-0.215 \pm 0.034$ |
| Predictive error | $0.0892 \pm 0.039$ | $0.0642 \pm 0.037$ | $0.065 \pm 0.038$ | $0.0592 \pm 0.03$ |
| Glass | Quadratic | Adaptive | Tilted | Probit |
| Marginal likelihood | $-319 \pm 5.6$ | $-193 \pm 3.9$ | $-193 \pm 5.4$ | $-201 \pm 2.6$ |
| Predictive likelihood | $-0.58 \pm 0.12$ | $-0.542 \pm 0.11$ | $-0.531 \pm 0.1$ | $-0.503 \pm 0.095$ |
| Predictive error | $0.197 \pm 0.032$ | $0.200 \pm 0.032$ | $0.200 \pm 0.032$ | $0.195 \pm 0.035$ |
| Thyroid | Quadratic | Adaptive | Tilted | Probit |
| Marginal likelihood | $-1814 \pm 43$ | $-909 \pm 30$ | $-916 \pm 31$ | $-840 \pm 18$ |
| Predictive likelihood | $-0.114 \pm 0.019$ | $-0.0793 \pm 0.014$ | $-0.0753 \pm 0.008$ | $-0.0916 \pm 0.010$ |
| Predictive error | $0.0241 \pm 0.0026$ | $0.0225 \pm 0.0024$ | $0.0226 \pm 0.0023$ | $0.0276 \pm 0.0028$ |

Table 1: Average results and standard deviations on three UCI datasets, based on 16 random $50 : 50$ training-test splits. Adaptive and tilted use NCVMP, quadratic and probit use VMP.

see Table 1. Here we have also included "Probit", corresponding to a Bayesian multinomial probit regression model, estimated using VMP, and similar in setup to [6], except that we use EP to approximate the predictive distribution, rather than sampling. On all three datasets the marginal likelihood calculated using the tilted or adaptive bounds is optimal out of the logistic models ("Probit" has a different underlying model, so differences in marginal likelihood are confounded by the Bayes factor). In terms of predictive performance the quadratic bound seems to be slightly worse across the datasets, with the performance of the other methods varying between datasets. We did not compare to the log bound since it is dominated by the tilted bound and is considerably slower to converge.

# 7 Discussion

NCVMP is not guaranteed to converge. Indeed, for some models we have found convergence to be a problem, which can be alleviated by damping: if the NCVMP message is $m_{f \to i}(x_i)$ then send the message $m_{f \to i}(x_i)^{1-\alpha} m_{f \to i}^{\text{old}}(x_i)^{\alpha}$ where $m_{f \to i}^{\text{old}}(x_i)$ was the previous message sent to $i$ and $0 \le \alpha < 1$ is a damping factor. The fixed points of the algorithm remained unchanged.

We have introduced Non-conjugate Variational Message Passing, which extends variational Bayes to non-conjugate models while maintaining the convenient message passing framework of VMP and allowing freedom to choose the most accurate available method to approximate required expectations. Deterministic and stochastic factors can be combined in a modular fashion, and conjugate parts of the model can be handled with standard VMP. We have shown NCVMP to be of practical use for fitting Bayesian binary and multinomial logistic models. We derived a new bound for the softmax integral which is tighter than other commonly used bounds, but has variational parameters that are still simple to optimise. Tightness of the bound is valuable both in terms of better approximating the posterior and giving a closer approximation to the marginal likelihood, which may be of interest for model selection.

# References

[1] H. Attias. A variational Bayesian framework for graphical models. *Advances in neural information processing systems*, 12(1-2):209215, 2000.

[2] M. Beal and Z. Ghahramani. Variational Bayesian learning of directed graphical models with hidden variables. *Bayesian Analysis*, 1(4):793832, 2006.

[3] D. Blei and J. Lafferty. A correlated topic model of science. *Annals of Applied Statistics*, 2007.

[4] D. Bohning. Multinomial logistic regression algorithm. *Annals of the Institute of Statistical Mathematics*, 44:197–200, 1992. 10.1007/BF00048682.

[5] G. Bouchard. Efficient bounds for the softmax and applications to approximate inference in hybrid models. In *NIPS workshop on approximate inference in hybrid models*, 2007.

[6] M. Girolami and S. Rogers. Variational bayesian multinomial probit regression with gaussian process priors. *Neural Computation*, 18(8):1790–1817, 2006.

[7] A. Honkela, T. Raiko, M. Kuusela, M. Tornio, and J. Karhunen. Approximate riemannian conjugate gradient learning for fixed-form variational bayes. *Journal of Machine Learning Research*, 11:3235–3268, 2010.

[8] A. Honkela, M. Tornio, T. Raiko, and J. Karhunen. Natural conjugate gradient in variational inference. In M. Ishikawa, K. Doya, H. Miyamoto, and T. Yamakawa, editors, *ICONIP (2)*, volume 4985 of *Lecture Notes in Computer Science*, pages 305–314. Springer, 2007.

[9] T. S. Jaakkola and M. I. Jordan. A variational approach to bayesian logistic regression models and their extensions. In *International Conference on Artificial Intelligence and Statistics*, 1996.

[10] M. E. Khan, B. M. Marlin, G. Bouchard, and K. P. Murphy. Variational bounds for mixed-data factor analysis. In *Advances in Neural Information Processing (NIPS) 23*, 2010.

[11] B. M. Marlin, M. E. Khan, and K. P. Murphy. Piecewise bounds for estimating bernoulli-logistic latent gaussian models. In *Proceedings of the 28th Annual International Conference on Machine Learning*, 2011.

[12] T. P. Minka. Expectation propagation for approximate bayesian inference. In *Uncertainty in Artificial Intelligence*, volume 17, 2001.

[13] T. P. Minka, J. M. Winn, J. P. Guiver, and D. A. Knowles. Infer.NET 2.4, 2010. Microsoft Research Cambridge. http://research.microsoft.com/infernet.

[14] H. Nickisch and C. E. Rasmussen. Approximations for binary gaussian process classification. *Journal of Machine Learning Research*, 9:2035–2078, Oct. 2008.

[15] M. Opper and C. Archambeau. The variational gaussian approximation revisited. *Neural Computation*, 21(3):786–792, 2009.

[16] Y. A. Qi and T. Jaakkola. Parameter expanded variational bayesian methods. In B. Schölkopf, J. C. Platt, and T. Hoffman, editors, *Advances in Neural Information Processing (NIPS) 19*, pages 1097–1104. MIT Press, 2006.

[17] T. Raiko, H. Valpola, M. Harva, and J. Karhunen. Building blocks for variational bayesian learning of latent variable models. *Journal of Machine Learning Research*, 8:155–201, 2007.

[18] L. K. Saul and M. I. Jordan. A mean field learning algorithm for unsupervised neural networks. *Learning in graphical models*, 1999.

[19] M. P. Wand, J. T. Ormerod, S. A. Padoan, and R. Fruhwirth. Variational bayes for elaborate distributions. In *Workshop on Recent Advances in Bayesian Computation*, 2010.

[20] J. Winn and C. M. Bishop. Variational message passing. *Journal of Machine Learning Research*, 6(1):661, 2006.

